# A 1,000-Neuron System with One Million 7-bit Physical Interconnections

**Yuzo Hirai**
Institute of Information Sciences and Electronics
University of Tsukuba
1-1-1 Ten-nodai, Tsukuba, Ibaraki 305, Japan
e-mail: hirai@is.tsukuba.ac.jp

## Abstract

An asynchronous PDM (Pulse-Density-Modulating) digital neural network system has been developed in our laboratory. It consists of one thousand neurons that are physically interconnected via one million 7-bit synapses. It can solve one thousand simultaneous nonlinear first-order differential equations in a fully parallel and continuous fashion. The performance of this system was measured by a winner-take-all network with one thousand neurons. Although the magnitude of the input and network parameters were identical for each competing neuron, one of them won in 6 milliseconds. This processing speed amounts to 360 billion connections per second. A broad range of neural networks including spatiotemporal filtering, feedforward, and feedback networks can be run by loading appropriate network parameters from a host system.

## 1 INTRODUCTION

The hardware implementation of neural networks is crucial in order to realize the real-time operation of neural functions such as spatiotemporal filtering, learning and constraint processings. Since the mid eighties, many VLSI chips and systems have been reported in the literature, e.g. [1] [2]. Most of the chips and the systems including analog and digital implementations, however, have focused on *feedforward* neural networks. Little attention has been paid to the dynamical aspect of *feedback* neural networks, which is especially important in order to realize constraint processings, e.g. [3]. Although there were a small number of exceptions that used analog circuits [4] [5], their network sizes were limited as compared to those of their feedforward counterparts because of wiring problems that are inevitable in regard to full and physical interconnections. To relax this problem, a pulse-stream system has been used in analog [6] and digital implementations [7].

The author developed a fully interconnected 54-neuron system that uses an asynchronous PDM (Pulse-Density-Modulating) digital circuit system [8]. The present paper describes a thousand-neuron system in which all of the neurons are physically interconnected via one million 7-bit synapses in order to create a fully parallel feedback system. The outline of this project was described in [10]. In addition to the enlargement of system size, synapse circuits were improved and time constant of each neuron was made variable. The PDM system was used because it can accomplish faithful analog data transmission between neurons and can relax wiring problems. An asynchronous digital circuit was used because it can solve scaling problems, and we could also use it to connect more than one thousand VLSI chips, as described below.

## 2   NEURON MODEL AND THE CIRCUITS

### 2.1   SINGLE NEURON MODEL

The behavior of each neuron in the system can be described by the following nonlinear first-order differential equation:

$$\mu_i \frac{dy_i^*(t)}{dt} = -y_i^*(t) + \sum_{j=1}^{N} w_{ij} y_j(t) + I_i(t), \tag{1}$$

$$y_i(t) = \varphi[y_i^*(t)], \text{ and} \tag{2}$$

$$\varphi[a] = \begin{cases} a & \text{if } a > 0 \\ 0 & \text{otherwise,} \end{cases} \tag{3}$$

where $\mu_i$ is a time constant of the $i$-th neuron, $y_i^*(t)$ is an internal potential of the $i$-th neuron at time $t$, $w_{ij}$ is a synaptic weight from the $j$-th to the $i$-th neurons, and $I_i(t)$ is an external input to the $i$-th neuron. $\varphi[a]$ is an analog threshold output function which becomes saturated at a given maximum value.

The system solves Eq.(1) in the following integral form:

$$y_i^*(t) = \int_0^t \left\{ -y_i^*(\tau) + \sum_{j=1}^{N} w_{ij} y_j(\tau) + I_i(\tau) \right\} \frac{d\tau}{\mu_i} + y_i^*(0), \tag{4}$$

where $y_i^*(0)$ is an initial value. An analog output of a neuron is expressed by a pulse stream whose frequency is proportional to the positive, instantaneous internal potential.

### 2.2   SINGLE NEURON CIRCUIT

#### 2.2.1   Synapse circuits

The circuit diagrams for a single neuron are shown in Fig. 1. As shown in Fig.1(a), it consists of synapse circuits, excitatory and inhibitory dendrite OR circuits, and a cell body circuit. Each synapse circuit transforms the instantaneous frequency of the input pulses to a frequency that is proportional to the synaptic weight. This transformation is carried out by a 6-bit rate multiplier, as shown in Fig.1(b). The behavior of a rate multiplier is illustrated in Fig.1(c) using a 3-bit case for brevity. A rate multiplier is a counter and its state transits to the next state when an input pulse occurs. Each binary bit of a given weight specifies at which states the output pulses are generated. When the LSB is on, an output pulse is generated at the fourth state. When the second bit is on, output pulses are generated at the second

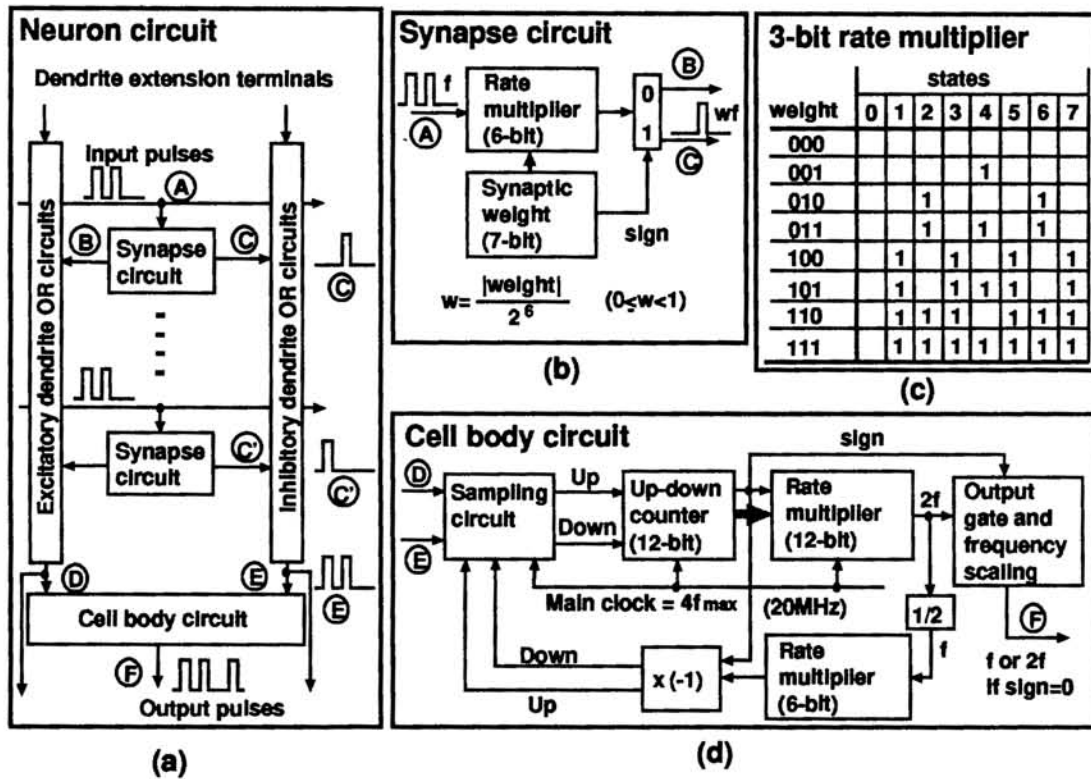

**3-bit rate multiplier**

| weight | 0 | 1 | 2 | 3 | 4 | 5 | 6 | 7 |
|---|---|---|---|---|---|---|---|---|
| 000 | | | | | | | | |
| 001 | | | | | 1 | | | |
| 010 | | | 1 | | | | 1 | |
| 011 | | | 1 | | 1 | | 1 | |
| 100 | | 1 | | 1 | | 1 | | 1 |
| 101 | | 1 | | 1 | 1 | 1 | | 1 |
| 110 | | 1 | 1 | 1 | | 1 | 1 | 1 |
| 111 | | 1 | 1 | 1 | 1 | 1 | 1 | 1 |

Figure 1: Circuit diagram of a single neuron. (a) Circuit diagram of a single neuron and (b) that of a synapse circuit. (c) To illustrate the function of a rate multiplier, the multiplication table for a 3-bit case is shown. (d) Circuit diagram of a cell body circuit. See details in text.

and at the sixth states. When the MSB is on, they are generated at all of the odd states. Therefore, the magnitude of synaptic weight that can be represented by a rate multiplier is less than one. In our circuit, this limitation was overcome by increasing the frequency of a neuron output by a factor of two, as described below.

### 2.2.2 Dendrite circuits

Output pulses from a synapse circuit are fed either to an excitatory dendrite OR circuit or to an inhibitory one, according to the synaptic weight sign. In each dendrite OR circuit, the synaptic output pulses are summed by OR gates, as is shown along the right side of Fig.1(a). Therefore, if these output pulses are synchronized, they are counted as one pulse and linear summation cannot take place. In our circuit, each neuron is driven by an individual clock oscillator. Therefore, they will tend to become desynchronized. The summation characteristic was analysed in [9], and it was shown to have a saturation characteristic that is similar to the positive part of a hyperbolic tangent function.

### 2.2.3 Cell body circuit

A cell body circuit performs the integration given by Eq.(4) as follows. As shown in Fig.1(d), integration was performed by a 12-bit up-down counter. Input pulses from an excitatory dendrite OR circuit are fed into the up-input of the counter and those from an inhibitory one are fed into the down-input after conflicts between

excitatory and inhibitory pulses have been resolved by a sampling circuit. A 12-bit rate multiplier produces internal pulses whose frequency is $2f$, where $f$ is proportional to the absolute value of the counter. The rate multiplier is driven by a main clock whose frequency is $4f_{max}$, $f_{max}$ being the maximum output frequency. When the counter value is positive, an output pulse train whose frequency is either $f$ or $2f$, according to the scale factor is transmitted from a cell body circuit.

The negative feedback term that appeared in the integrand of Eq.(4) can be realized by feeding the internal pulses into the down-input of the counter when the counter value is positive and feeding them into the up-input when it is negative. The 6-bit rate multiplier inserted in this feedback path changes the time constant of a neuron. Let $\beta_i$ be the rate value of the rate multiplier, where $0 \le \beta_i < 2^6$. The Eq.(4) becomes:

$$
\begin{aligned}
y_i^*(t) &= \int_0^t \left\{ -\frac{\beta}{2^6} y_i^*(\tau) + \sum_{j=1}^{N} w_{ij} y_j(\tau) + I_i(\tau) \right\} \frac{d\tau}{\mu_i} + y_i^*(0) \\
&= \int_0^t \left\{ -y_i^*(\tau) + \frac{2^6}{\beta} \left( \sum_{j=1}^{N} w_{ij} y_j(\tau) + I_i(\tau) \right) \right\} \frac{d\tau}{\frac{\mu_i 2^6}{\beta}} + y_i^*(0). \quad (5)
\end{aligned}
$$

Therefore, the time constant changes to $\frac{\mu_i 2^6}{\beta}$, where $\mu_i$ was given by $\frac{2^{11}}{f_{max}}$ seconds. It should be noted that, since the magnitude of the total input was increased by a factor of $\frac{2^6}{\beta}$, the strength of the input should be decreased by the inverse of that factor in order to maintain an appropriate output level. If it is not adjusted, we can increase the input strength. Therefore, the system has both input and output scaling functions. The time constant varies from about $416\mu sec$ for $\beta = 63$ to $26.2 msec$ for $\beta = 1$. When $\beta = 0$, the negative feedback path is interrupted and the circuit operates as a simple integrator, and every feedforward network can be run in this mode of operation.

## 3   THE 1,000-NEURON SYSTEM

### 3.1   VLSI CHIP

A single type of VLSI chip was fabricated using a $0.7\mu m$ CMOS gate array with 250,000 gates. A single chip contains 18 neurons and 51 synapses for each neuron. Therefore, each chip has a total of 918 synapses. About 85% of the gates in a gate array could be used, which was an extremely efficient value. A chip was mounted on a flat package with 256 pins. Among them, 216 pins were used for signals and the others were used for twenty pairs of $V_{CC}(=3.3V)$ and GND.

### 3.2   THE SYSTEM

As illustrated in Fig.2(a), this system consists of $56 \times 20 = 1,120$ chips. 56 chips are used for both cell bodies and synapses, and the others are used to extend dendrite circuits and increase the number of synapses. In order to extend the dendrites, the dendrite signals in a chip can be directly transmitted to the dendrite extention terminals of another chip by bypassing the cell body circuits. There are $51 \times 20 = 1,020$ synapses per neuron. Among them, 1,008 synapses are used for fully hard-wired interconnections and the other 12 synapses are used to receive external signals. There are a total of 1,028,160 synapses in this system. It is controlled by a personal computer. The synaptic weights, the contents of the up-down counters

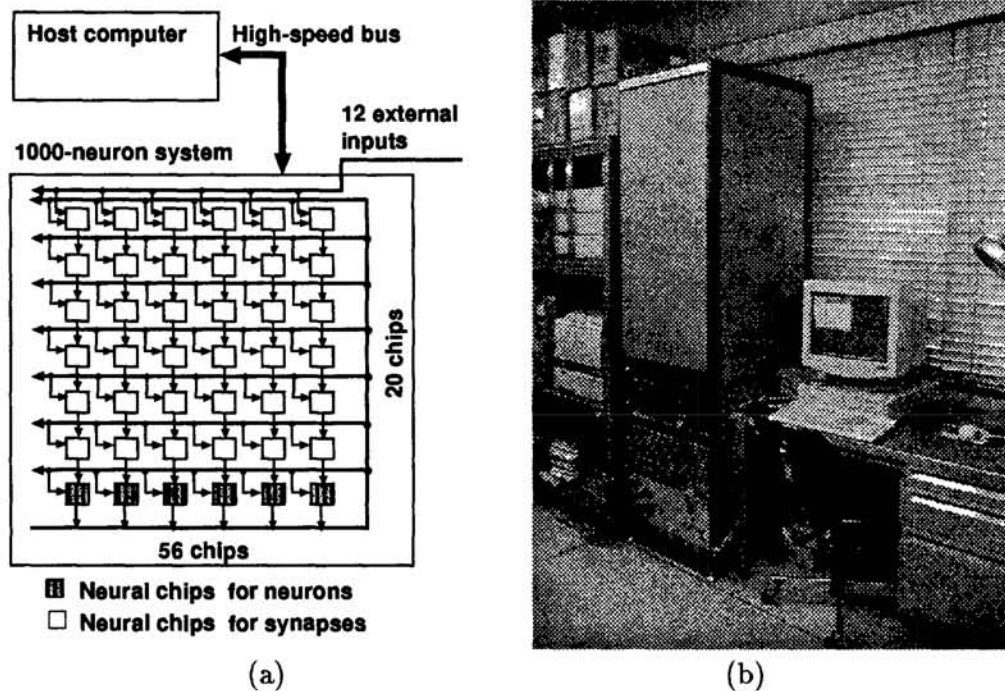

Figure 2: Structure of the system. (a) System configuration. The down arrows emitted from the open squares designate signal lines that are extending dendrites. The others designate neuron outputs. (b) Exterior of the system. It is controlled by a personal computer.

and the control registers can be read and written by the host system. It takes about 6 seconds to set all the network parameters from the host system.

The exterior of this system is shown in Fig.2(b). Inside the cabinet, there are four shelves. In each shelf, fourteen circuit boards were mounted and on each board 20 chips were mounted. One chip was used for 18 neurons and the other chips were used to extend the dendrites. Each neuron is driven by an individual 20MHz clock oscillator.

## 4   SYSTEM PERFORMANCE

In order to measure the performance of this system, one neuron was used as a signal generator. By setting all the synaptic weights and the internal feedback gain of a signal neuron to zero, and by setting the content of the up-down counter to a given value, it can produce an output with a constant frequency that is proportional to the counter value. The input strength of the other neurons can be adjusted by changing the counter value of a signal neuron or the synaptic weights from it.

The step reponses of a neuron to different inputs are shown in Fig.3(a). As seen in the figure, the responses exactly followed Eq.(1) and the time constant was about 400$\mu$sec. Figure 3(b) shows responses with different time constants. The inputs were identical for all cases.

Figure 3(c) shows the response of a temporal filter that was obtained by the difference between a fast and a slow neuron. By combining two low-pass filters that had different cutoff frequencies, a band-pass filter was created. A variety of spatiotem-

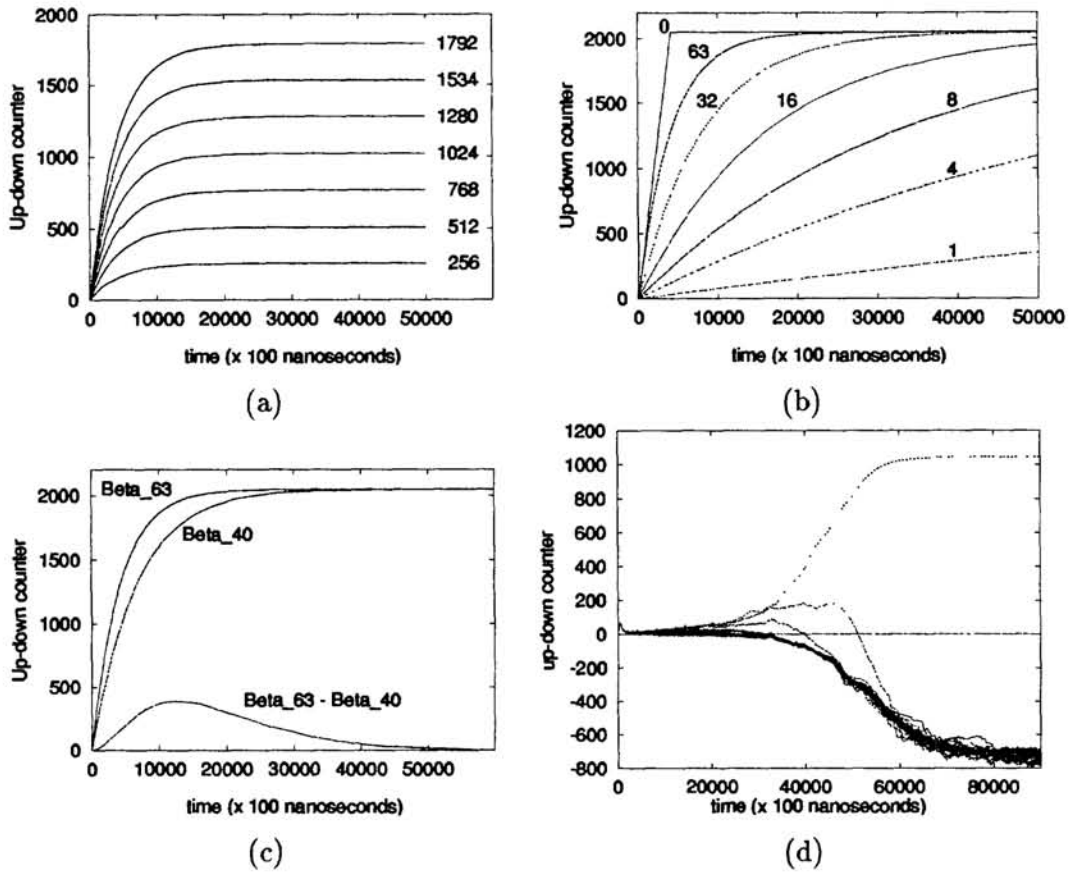

Figure 3: Responses obtained by the system. (a) Step responses to different input levels. Parameters are the values that are set in the up-down counter of a signal neuron. (b) Step responses for different time constants. Parameters are the values of $\beta_i$ in Eq.5. Inputs were identical in all cases. (c) Response of a temporal filter that was obtaind by the difference between a fast and a slow neuron. (d) Response of a winner-take-all network among 1,007 neurons. The responses of a winner neuron and 24 of the 1,006 defeated neurons are shown.

poral filters can be implemented in this way.

Figure 3(d) shows the responses of a winner-take-all network among 1,007 neurons. The time courses of the responses of a winner neuron and 24 of the 1,006 defeated neurons are shown in the figure. The strength of all of the inhibitory synaptic weights between neurons was set to $2 \times \left(-\frac{48}{64}\right)$, where 2 is an output scale factor. The synaptic weights from a signal neuron to the 1,007 competing ones were identical and were $\frac{32}{64}$. Although the network parameters and the inputs to all competing neurons were identical, one of them won in 6 msec. Since the system operates asynchronously and the spatial summation of the synaptic output pulses is probabilistic, one of the competing neurons can win in a stochastic manner.

In order to derive the processing speed in terms of *connections per second*, the same winner-take-all network was solved by the Euler method on a latest workstation. Since it took about 76.2 seconds and 2,736 iterations to converge, the processing speed of the workstation was about 36 million connections per second $\left(\approx \frac{1,007 \times 1,007 \times 2,736}{76.2}\right)$. Since this system is 10,000 times faster than the workstation,

the processing speed amounts to 360 billion connections per second.

Various kinds of neural networks including spatiotemporal filtering, feedforward and feedback neural networks can be run in this single system by loading appropriate network parameters from the host system. The second version of this system, which can be used via the Internet, will be completed by the end of March, 1998.

**Acknowledgements**

The author is grateful to Mr. Y. Kuwabara and Mr. T. Ochiai of Hitachi Microcomputer System Ltd. for their collaboration in developing this system and to Dr. M. Yasunaga and Mr. M. Takahashi for their help in testing it. The author is also grateful to Mr. H. Toda for his collaboration in measuring response data. This work was supported by "Proposal-Based Advanced Industrial Technology R&D Program" from NEDO in Japan.

# References

[1] C. Mead: *Analog VLSI and Neural Systems.* Addison-Wesley Publishing Company, Massachusetts, 1989

[2] K.W.Przytula and V.K.Prasanna, Eds.: *Parallel Digital Implementations of Neural Networks.* Prentice Hall, New Jersey, 1993

[3] J.J. Hopfield: Neurons with graded response have collective computational properties like those of two-state neurons. *Proc. Natl. Acad. Sci. U.S.A.*, **81**, pp.3088-3092, 1984

[4] P. Mueller, J. van der Spiegel, V. Agami, D. Blackman, P. Chance, C. Donham, R. Etienne, J. Kim. M. Massa and S. Samarasekera: Design and performance of a prototype analog neural computer. *Proc. the 2nd International Conf. on Microelectronics for Neural Networks*, pp.347-357, 1991

[5] G. Cauwenberghs: A learning analog neural network chip with continuous-time recurrent dynamics. In J. D. Cowan, G. Tesauro and J. Alspector, Eds., *Advances in Neural Information Processing Systems 6*, Morgan Kaufmann Publishers, San Mateo, CA, pp.858-865, 1994

[6] S. Churcher, D. J. Baxter, A. Hamilton, A. F. Murry, and H. M. Reekie: Generic analog neural computation – The EPSILON chip. In S. J. Hanson, J. D. Cowan and C. L. Giles, Eds., *Advances in Neural Information Processing Systems 6*, Morgan Kaufmann Publishers, San Mateo, CA, pp.773-780, 1993

[7] H. Eguchi, T. Furuta, H. Horiguchi, S. Oteki and T. Kitaguchi: Neural network LSI chip with on-chip learning. *Proceedings of IJCNN'91 Seattle*, Vol.I/453-456, 1991

[8] Y. Hirai, et al.: A digital neuro-chip with unlimited connectability for large scale neural networks. *Proc. International Joint Conf. on Neural Networks'89 Washington D.C.*, Vol.II/163-169, 1989

[9] Y.Hirai, *VLSI Neural Network Systems* (Gordon and Breach Science Publishers, Birkshire, 1992)

[10] Y. Hirai and M. Yasunaga: A PDM digital neural network system with 1,000 neurons fully interconnected via 1,000,000 6-bit synapses. *Proc. International Conference on Neural Information Processing'96*, Vol.II/1251, 1996